# Performance Through Consistency: MS-TDNN's for Large Vocabulary Continuous Speech Recognition

**Joe Tebelskis and Alex Waibel**
School of Computer Science
Carnegie Mellon University
Pittsburgh, PA 15213

## Abstract

Connectionist speech recognition systems are often handicapped by an inconsistency between training and testing criteria. This problem is addressed by the Multi-State Time Delay Neural Network (MS-TDNN), a hierarchical phoneme and word classifier which uses DTW to modulate its connectivity pattern, and which is directly trained on word-level targets. The consistent use of word accuracy as a criterion during both training and testing leads to very high system performance, even with limited training data. Until now, the MS-TDNN has been applied primarily to small vocabulary recognition and word spotting tasks. In this paper we apply the architecture to large vocabulary continuous speech recognition, and demonstrate that our MS-TDNN outperforms all other systems that have been tested on the CMU Conference Registration database.

## 1 INTRODUCTION

Neural networks hold great promise in the area of speech recognition. But in order to fulfill their promise, they must be used "properly". One obvious condition of "proper" use is that both training and testing should use a consistent error criterion. Unfortunately, in speech recognition, this obvious condition is often violated: networks are frequently trained using phoneme-level criteria (phoneme classification

or acoustic prediction), while the testing criterion is word recognition accuracy. If phoneme recognition were perfect, then word recognition would also be perfect; but of course this is not the case, and the errors which are inevitably made are optimized for the wrong criterion, resulting in suboptimal word recognition accuracy.

The Multi-State Time Delay Neural Network (MS-TDNN) has recently been proposed as a solution to this problem [1]. The MS-TDNN is a hierarchically structured classifier which consistently uses word accuracy as a criterion for both training and testing. It has so far yielded excellent results on small vocabulary recognition [1, 2, 3] and a word spotting task [4]. In the present paper, we review the MS-TDNN architecture, discuss its application to large vocabulary continuous speech recognition, and present some favorable experimental results on this task.

## 2   RESOLVING INCONSISTENCIES: MS-TDNN ARCHITECTURE

In this section we motivate the design of our MS-TDNN by showing how a series of intermediate designs resolve successive inconsistencies.

The preliminary system architecture, shown in Figure 1(a), simply consists of a phoneme classifier, in this case a TDNN, whose outputs are copied into a DTW matrix, in which continuous speech recognition is performed. Many existing systems are based on this type of approach. While this is fine for bootstrapping purposes, the design is ultimately suboptimal because the training criterion is inconsistent with the testing criterion: phoneme classification is not word classification.

To address this inconsistency we must train the network explicitly to perform word classification. To this end, we define a word layer with one unit for each word in the vocabulary; the idea is illustrated in Figure 1(b) for the word "cat". We correlate the activation of the word unit with the associated DTW score by establishing connections from the DTW alignment path to the word unit. Also, we give the phonemes within a word independently trainable weights, to enhance word discrimination (for example, to discriminate "cat" from "mat" it may be useful to give special emphasis to the first phoneme); these weights are tied over all frames in which the phoneme occurs. Thus a word unit is an ordinary unit, except that its connectivity to the preceding layer is determined dynamically, and its net input should be normalized by the total duration of the word. The word unit is trained on a target of 1 or 0, depending if the word is correct or incorrect for the current segment of speech, and the resulting error is backpropagated through the entire network. Thus, word discrimination is treated very much like phoneme discrimination.

Although network (b) resolves the original inconsistency, it now suffers from a secondary one – namely, that the weights leading to a word unit are used during training but ignored during testing, since DTW is still performed entirely in the DTW layer. We resolve this inconsistency by "pushing down" these weights one level, as shown in Figure 1(c). Now the phoneme activations are no longer directly copied into the DTW layer, but instead are modulated by a weight and bias before being stored there (DTW units are linear); and the word unit has constant weights, and no bias. During word-level training, error is still backpropagated from targets at the word level, but biases and weights are modified only at the DTW level and

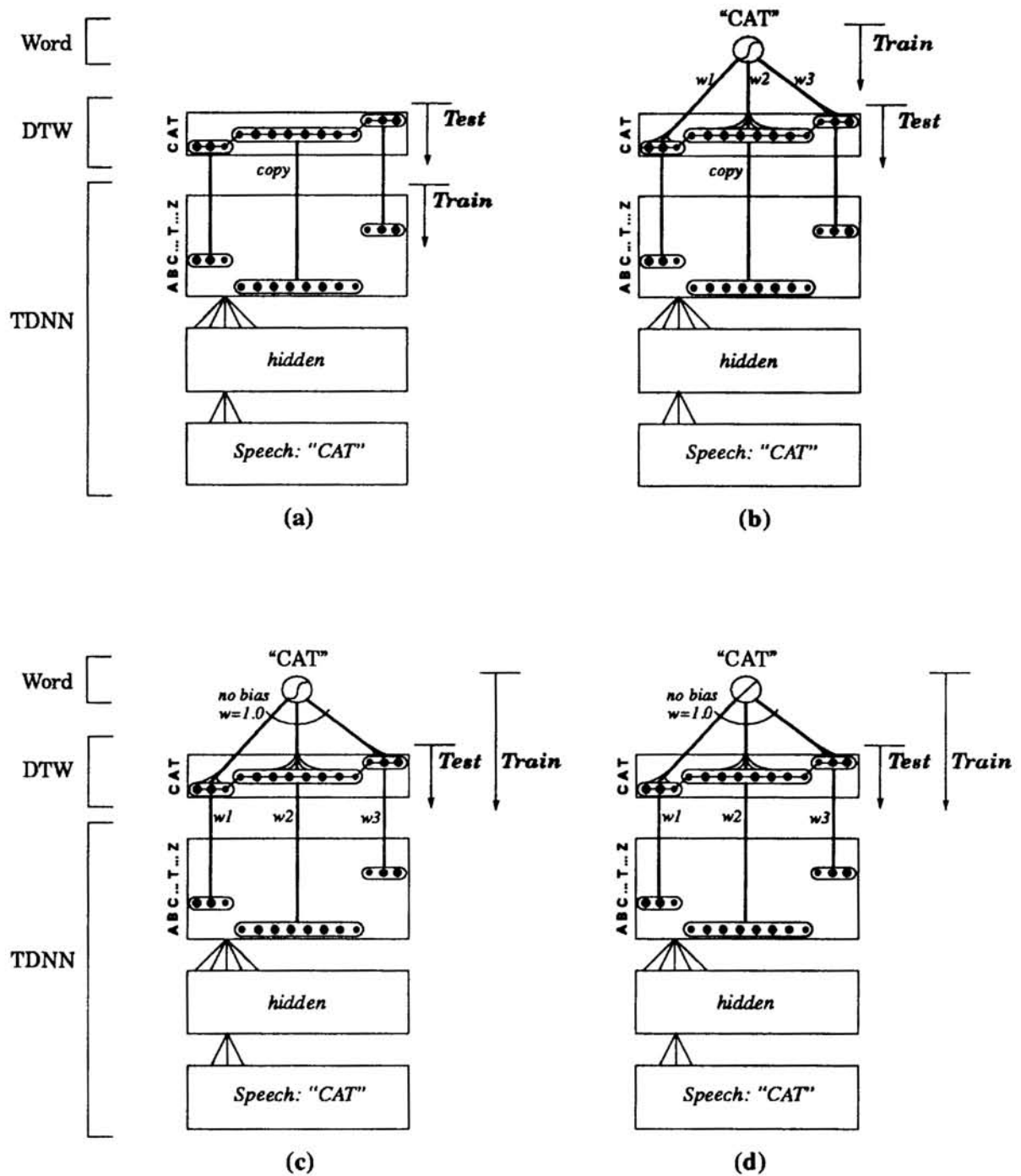

Figure 1: Resolving inconsistencies: (a) TDNN+DTW. (b) Adding word layer.
(c) Pushing down weights. (d) Linear word units, for continuous speech recognition.

below. Note that this transformed network is not exactly equivalent to the previous one, but it preserves the properties that there are separate learned weights associated with each phoneme, and there is an effective bias for each word.

Network (c) is still flawed by a minor inconsistency, arising from its sigmoidal word unit. The problem does not exist for isolated word recognition, since any monotonic function (sigmoidal or otherwise) will correlate the highest word activation with the highest DTW score. However, for continuous speech recognition, which concatenates words into a sentence, the optimal sum of sigmoids may not correspond to the optimal sigmoid of a sum, leading to an inconsistency between word and sentence recognition. Linear word units, as shown in (d), resolve this problem; in practice we have found that linear word units perform slightly better than sigmoidal word units.

The resulting architecture is called a "Multi-State TDNN" because it integrates the DTW alignment of multiple states into a TDNN to perform word classification. While an MS-TDNN for small vocabulary recognition can be based on word models with non-shared states [4], a large vocabulary MS-TDNN must be based on shared units of speech, such as phonemes. In our system, the TDNN (first three layers) is shared by all words in the vocabulary, while each word requires only one non-shared weight and bias for each of its phonemes. Thus the number of parameters in the MS-TDNN remains moderate even for a large vocabulary, and it can make the most of limited training data. Moreover, new words can be added to the vocabulary without retraining, by simply defining a new DTW layer for each new word, with incoming weights and biases initialized to 1.0 and 0.0, respectively.

Given constant weights under the word layer, word level training is really just another way of viewing DTW level training; but the former is conceptually simpler because there is a single binary target for each word, which makes word level discrimination very straightforward. For a large vocabulary, discriminating against all incorrect words would be very expensive, so we discriminate against only a small number of close matches (typically 1).

Word level training yields better word classification than phoneme level training. In one experiment, for example, we bootstrapped our system with phoneme level training (as shown in Figure 1a), and found that word recognition accuracy asymptoted at 71% on a test set. We then continued training from the word level (as shown in Figure 1d), and found that word accuracy improved to 81% on the test set. It is worth noting that in an intermediate experiment, even when we held the DTW layer's incoming weights and biases constant (at 1.0 and 0.0 respectively), thus adding no new trainable parameters to the system, we found that word level training still improved the word accuracy from 71% to 75% on the test set, as a consequence of word-level discrimination.

## 3   BALANCING THE TRAINING SET

The MS-TDNN must be first bootstrapped with phoneme level training. In our early experiments we had difficulty bootstrapping the TDNN, not only because our training set was unbalanced, but also because the vast majority of phonemes were being trained on a target of 0, so that the negative training was overwhelming and

|   | N | O | N | A | T | Y | E | T |
|---|---|---|---|---|---|---|---|---|
| A | ○ | ○ | ○ | ● | ○ | ○ | ○ | ○ |
| E | ○ | ○ | ○ | ○ | ○ | ○ | ● | ○ |
| O | ○ | ○ | ● | ○ | ○ | ○ | ○ | ○ |
| N | ● | ○ | ● | ○ | ○ | ○ | ○ | ○ |
| T | ○ | ○ | ○ | ○ | ● | ○ | ○ | ● |
| Y | ○ | ○ | ○ | ○ | ○ | ● | ○ | ○ |
| Z | ○ | ○ | ○ | ○ | ○ | ○ | ○ | ○ |

**Scale backprop error by:**

|   | N | O | N | A | T | Y | E | T |
|---|---|---|---|---|---|---|---|---|
| A | -1/7 | -1/7 | -1/7 | **1.0** | -1/7 | -1/7 | -1/7 | -1/7 |
| E | -1/7 | -1/7 | -1/7 | -1/7 | -1/7 | -1/7 | **1.0** | -1/7 |
| O | . | . | . | . | | | | |
| N | **1/2** | -1/6 | **1/2** | -1/6 | -1/6 | -1/6 | -1/6 | -1/6 |
| T | -1/6 | -1/6 | -1/6 | -1/6 | **1/2** | -1/6 | -1/6 | **1/2** |
| Y | . | . | . | . | | | | |
| Z | -1/8 | -1/8 | -1/8 | -1/8 | -1/8 | -1/8 | -1/8 | -1/8 |

Figure 2: Balancing the training set: "No not yet".

defeating the positive training. In order to address these problems, we normalized the amount of error backpropagated from each phoneme unit so that the relative influence of positive and negative training was balanced out over the entire training set.

This apparently novel technique is illustrated in Figure 2. Given the utterance "No not yet", for example, we observe that there are two frames each of "N" and "T", one frame of several other phonemes, and zero of others. Based on these counts, we compute a backpropagation scaling factor for each phoneme in each frame, as shown in the bottom half of the figure.

We found that this technique was indispensible when bootstrapping with the squared error criterion, $E = \sum (T_i - Y_i)^2$. In subsequent experiments, we found that it was still somewhat helpful but no longer necessary when training with the McClelland error function, $E = -\sum log(1 - (T_i - Y_i)^2)$, or with the Cross Entropy error function, $E = -\sum (T_i log Y_i) + (1 - T_i) log(1 - Y_i)$. We attribute this difference to the fact that the sum squared error function is merely a quadratic error function, whereas the latter two functions tend towards infinite error as the difference between the target and actual activation approaches its maximum value, compensating more forcefully for the flat behavior encouraged by all the negative training.

## 4    EXPERIMENTAL RESULTS

We have trained and tested our MS-TDNN on two recordings of 200 English sentences from the CMU Conference Registration database, recorded by one male speaker using a close-speaking microphone. Our speaker-dependent testing results

Table 1: Comparison of speech recognition systems applied to the CMU Conference Registration Database. HMM-n = HMM with n mixture densities [5]. LPNN = Linked Predictive Neural Network [6]. HCNN = Hidden Control Neural Network [7]. LVQ = Learned Vector Quantization [8]. TDNN corresponds to MS-TDNN without word-level training. (Perplexity 402(a) used only 41 test sentences; 402(b) used 204 test sentences.)

| System | perplexity | | | |
|--------|-----|------|--------|--------|
|        | 7   | 111  | 402(a) | 402(b) |
| HMM-1  |     | 55%  |        |        |
| HMM-5  | 96% | 71%  | 58%    |        |
| HMM-10 | 97% | 75%  | 66%    |        |
| LPNN   | 97% | 60%  | 41%    |        |
| HCNN   |     | 75%  |        |        |
| LVQ    | 98% | 84%  | 74%    | 61%    |
| TDNN   | 98% | 78%  | 72%    | 64%    |
| **MS-TDNN** | **98%** | **82%** | **81%** | **70%** |

are given in Table 1, along with comparative results from several other systems. It can be seen that the MS-TDNN has outperformed all other systems that have been compared on this database. This particular MS-TDNN used 16 melscale spectral input units, 20 hidden units, 120 phoneme units (40 phonemes with a 3-state model), 5,487 DTW units, and 402 word units, with 3 and 5 delays respectively in the first two layers of weights, giving a total of 24,074 weights; it used symmetric (-1..1) unit activations and inputs, and linear DTW units and word units. Word level training was performed using the Classification Figure of Merit (CFM) error function, $E = (1 + (Y_{\bar{c}} - Y_c))^2$, in which the correct word (with activation $Y_c$) is explicitly discriminated from the best incorrect word (with activation $Y_{\bar{c}}$); CFM was somewhat better than MSE for word level training, although the opposite was true for phoneme level training. Negative word level training was performed only if the two words were sufficiently confusable, in order to avoid disrupting the network on behalf of words that had already been well-learned.

More recently we have begun experiments on the speaker independent Resource Management database, containing nearly 4000 training sentences. To date we have primarily focused on bootstrapping the phoneme level TDNN on this database, without doing much word level training; but early experiments suggest we may reasonably expect another 4% improvement from word level training. Our preliminary results are shown in Table 2, compared against two other systems: an early version of Sphinx [9], and a simple but large MLP which has been trained as a phoneme classifier [10]. Each of these systems uses context independent phoneme models, with multiple states per phoneme, and includes differenced coefficients in its input representation. It can be seen that our TDNN outperforms this version of Sphinx while using a comparable number of parameters, but is outperformed by the MLP which has an order of magnitude more parameters. (We note that the MLP also uses phonological models to enhance its performance, and uses online training with ran-

Table 2: Context-independent systems applied to the Resource Management database.

| System | parameters | perplexity | |
| --- | --- | --- | --- |
| | | 60 | 1000 |
| Early Sphinx | 35,000 | 76% | 36% |
| MLP (ICSI-SRI) | 300,000 | 94% | 75% |
| TDNN | 42,000 | 79% | 43% |
| MS-TDNN | 75,000 | (+4%) | - |

dom sampling rather than updating the weights after each sentence.) In any case, we suggest that the MLP might further improve its performance by incorporating word level training.

## 5  REMAINING INCONSISTENCIES

While the MS-TDNN was designed for consistency, it is not yet entirely consistent. For example, the MS-TDNN's training algorithm assumes that the network connectivity is fixed; but in fact the connectivity at the word level varies, depending on the DTW alignment path during the current iteration. We presume that this is a negligible factor, however, by the time the training has asymptoted and the segmentation has stabilized.

A more serious inconsistency arises during discriminative training. In our MS-TDNN, negative training is performed at known word boundaries; this is inconsistent because word boundaries are in fact unknown during testing. It would be better to discriminate against words found by a free alignment, as suggested by Hild [3]. Unfortunately this is an expensive operation, and it proved impractical for our system.

## 6  CONCLUSION

We have shown that the performance of a connectionist speech recognition system can be improved by resolving inconsistencies in its design. Specifically, by introducing word level training into a TDNN phoneme classifier (thus defining an MS-TDNN), the training and testing criteria become consistent, enhancing the system's word recognition accuracy. We applied our MS-TDNN architecture to the task of large vocabulary continuous speech recognition, and found that it outperforms all other systems that have been evaluated on the CMU Conference Registration database. In addition, preliminary results suggest that the MS-TDNN may perform well on the large vocabulary Resource Management database, using a relatively small number of free parameters. Our future work will focus on this investigation.

## Acknowledgements

The authors gratefully acknowledge the support of DARPA and the National Science Foundation.

## References

[1] P. Haffner, M. Franzini, and A. Waibel. Integrating Time Alignment and Connectionist Networks for High Performance Continuous Speech Recognition. In *Proc. International Conference on Acoustics, Speech, and Signal Processing (ICASSP)*, 1991.

[2] P. Haffner. Connectionist Word-Level Classification in Speech Recognition. In *Proc. ICASSP*, 1992.

[3] H. Hild and A. Waibel. Connected Letter Recognition with a Multi-State Time Delay Neural Network. In *Advances in Neural Information Processing Systems 5*, Morgan Kaufmann Publishers, 1993.

[4] T. Zeppenfeld and A. Waibel. A Hybrid Neural Network, Dynamic Programming Word Spotter. In *Proc. ICASSP*, 1992.

[5] O. Schmidbauer. An LVQ Based Reference Model for Speaker-Independent and -Adaptive Speech Recognition. Technical Report, Carnegie Mellon University, 1991.

[6] J. Tebelskis, A. Waibel, B. Petek, and O. Schmidbauer. Continuous Speech Recognition using Linked Predictive Neural Networks. In *Proc. ICASSP*, 1991.

[7] B. Petek and J. Tebelskis. Context-Dependent Hidden Control Neural Network Architecture for Continuous Speech Recognition. In *Proc. ICASSP*, 1992.

[8] O. Schmidbauer and J. Tebelskis. An LVQ Based Reference Model for Speaker Adaptive Speech Recognition. In *Proc. ICASSP*, 1992.

[9] K. F. Lee. Large Vocabulary Speaker-Independent Continuous Speech Recognition: The SPHINX System. PhD Thesis, Carnegie Mellon University, 1988.

[10] M. Cohen, H. Franco, N. Morgan, D. Rumelhart, and V. Abrash. Context-Dependent Multiple Distribution Phonetic Modeling with MLP's. In *Advances in Neural Information Processing Systems 5*, Morgan Kaufmann Publishers, 1993.